# Convex Relaxation of Mixture Regression with Efficient Algorithms

**Novi Quadrianto, Tibério S. Caetano, John Lim**
NICTA - Australian National University
Canberra, Australia
{firstname.lastname}@nicta.com.au

**Dale Schuurmans**
University of Alberta
Edmonton, Canada
dale@cs.ualberta.ca

## Abstract

We develop a convex relaxation of maximum a posteriori estimation of a mixture of regression models. Although our relaxation involves a semidefinite matrix variable, we reformulate the problem to eliminate the need for general semidefinite programming. In particular, we provide two reformulations that admit fast algorithms. The first is a max-min spectral reformulation exploiting quasi-Newton descent. The second is a min-min reformulation consisting of fast alternating steps of closed-form updates. We evaluate the methods against Expectation-Maximization in a real problem of motion segmentation from video data.

## 1 Introduction

Regression is a foundational problem in machine learning and statistics. In practice, however, data is often better modeled by a *mixture* of regressors, as demonstrated by the prominence of mixture regression in a number of application areas. Gaffney and Smyth [1], for example, use mixture regression to cluster trajectories, i.e. sets of short sequences of data such as cyclone or object movements in video sequences as a function of time. Each trajectory is believed to have been generated from one of a number of components, where each component is associated with a regression model. Finney et al. [2] have employed an identical mixture regression model in the context of planning: regression functions are strategies for a given planning problem. Elsewhere, the mixture of regressors model has been shown to be useful in addressing covariate shift, i.e. the situation where the distribution of the training set used for modeling does not match the distribution of the test set in which the model will be used. Storkey and Sugiyama [3] model the covariate shift process in a mixture regression setting by assuming a shift in the mixing proportions of the components.

In each of these problems, one must estimate $k$ distinct latent regression functions; that is, estimate functions whose values correspond to the mean of response variables, under the assumption that the response variable is generated by a mixture of $k$ components. This estimation problem can be easily tackled if it is known to which component each response variable belongs (yielding $k$ independent regression problems). However in general the component of a given observation is not known and is modeled as a latent variable. A commonly adopted approach for maximum-likelihood estimation with latent variables (in this case, component membership for each response variable) is Expectation-Maximization (EM) [4]. Essentially, EM iterates inference over the hidden variables and parameter estimation of the resulting decoupled models until a *local* optimum is reached. We are not aware of any approach to maximum likelihood estimation of a mixture of regression models that is not based on the non-convex marginal likelihood objective of EM.

In this paper we present a convex relaxation of maximum a posteriori estimation of a mixture of regression models. Recently, convex relaxations have gained considerable attention in machine learning (c.f. [5, 6]). By exploiting convex duality, we reformulate a relaxation of mixture regression as a semidefinite program. To achieve a scalable approach, however, we propose two reformulations that admit fast algorithms. The first is a max-min optimization problem which can be solved by iterations of quasi-Newton steps and eigenvector computations. The second is a min-min optimization problem solvable by iterations of closed-form solutions. We present experimental results comparing our methods against EM, both in synthetic problems and real computer vision problems, and show some benefits of a convex approach over a local solution method.

**Related work** Goldfeld and Quandt [7] introduced a mixture regression model with two components called *switching regressions*. The problem is re-cast into a single composite regression equation by introducing a switching variable. A consistent estimator is then produced by a continuous relaxation of this switching variable. An EM algorithm for switching regressions was first presented by Hosmer [8]. Späth [9] introduced a problem called *clusterwise linear regression*, consisting of finding a $k$-partition of the data such that a least squares regression criterion within those partitions becomes a minimum. A non-probabilistic algorithm similar to $k$-means was proposed. Subsequently, the general $k$-partition case employing EM was developed (c.f. [10, 11, 1]) and extended to various situations including the use of variable length trajectory data and to non-parametric regression models. In the extreme, each individual could have its specific regression model but coupled at higher level with a mixture on regression parameters [12]. An EM algorithm is again employed to handle hidden data, in this case group membership of parameters. The Hierarchical Mixtures of Experts [13] model also shares some similarity to mixture regression in that gating networks which contain mixtures of generalized linear models are defined. In principle, our algorithmic advances can be applied to many of these formulations.

## 2 The Model

**Notation** In the following we use the uppercase letters $(X, \Pi, \Psi)$ to denote matrices and the lowercase letters $(x, y, w, \pi, \psi, c)$ to denote vectors. We use $t$ to denote the sample size, $n$ to denote the dimensionality of the data and $k$ to denote the number of mixture components. $\Lambda(a)$ denotes a diagonal matrix whose diagonal is equal to vector $a$, and $\text{diag}(A)$ is a vector equal to the diagonal of matrix $A$. Finally, we let $\mathbf{1}$ denote the vector of all ones, use $\odot$ to denote Hadamard (componentwise) matrix product, and use $\otimes$ to denote Kronecker product.

We are given a matrix of regressors $X \in \mathbb{R}^{t \times n}$ and a vector of regressands $y \in \mathbb{R}^{t \times 1}$ where the response variable $y$ is generated by a mixture of $k$ components, but we do not know which component of the mixture generates each response $y_i$. We therefore use the matrix $\Pi \in \{0, 1\}^{t \times k}, \Pi\mathbf{1} = \mathbf{1}$, to denote the hidden assignment of mixture labels to each observation: $\Pi_{ij} = 1$ iff observation $i$ has mixture label $j$. We use $x_i$ to denote the $i^{th}$ row of $X$ (i.e. observation $i$ as a row vector), $\pi_i$ to denote the $i^{th}$ row of $\Pi$ and $y_i$ to denote the $i^{th}$ element of $y$. We assume a linear generative model for $y_i$ on a feature representation $\psi_i = \pi_i \otimes x_i$, under i.i.d. sampling

$$y_i | x_i, \pi_i = \psi_i w + \epsilon_i, \quad \epsilon_i \sim N(0, \sigma^2), \tag{1}$$

where $w \in \mathbb{R}^{(n \times k) \times 1}$ is the vector of stacked parameter vectors of the components. We therefore have the likelihood

$$p(y_i | x_i, \pi_i; w) = \frac{1}{\sqrt{2\pi\sigma^2}} \exp\left[-\frac{1}{2\sigma^2}(\psi_i w - y_i)^2\right] \tag{2}$$

for a single observation $i$ (recalling that $\psi_i$ depends on both $x_i$ and $\pi_i$). We further impose a Gaussian prior on $w$ for capacity control. Also, one may want to constrain the size of the largest mixture component. For that purpose one could constrain the solutions $\Pi$ such that $\max(\text{diag}(\Pi^T\Pi)) \leq \gamma t$, where $\gamma t$ is an upper bound on the size of the largest component ($\gamma$ is an upper bound on the proportion of the largest component). Combining these assumptions and adopting matrix notation we obtain the optimization problem: minimize the negative log-posterior of the entire sample

$$\min_{\Pi, w}\left[\sum_i A(\psi_i, w) - \frac{1}{\sigma^2}y^T\Psi w + \frac{1}{2\sigma^2}y^T y + \frac{\alpha}{2}w^T w\right], \text{ where} \tag{3}$$

$$A(\psi_i, w) = \frac{1}{2\sigma^2}w^T\psi_i^T\psi_i w + \frac{1}{2}\log(2\pi\sigma^2). \tag{4}$$

Here $\Psi$ is the matrix whose rows are the vectors $\psi_i = \pi_i \otimes x_i$. Since $X$ is observed, note that the optimization only runs over $\Pi$ in $\Psi$. The constraint $\max(\text{diag}(\Pi^T\Pi)) \leq \gamma t$ may also be added.

Eliminating constant terms, our final task will be to solve

$$\min_{\Pi, w}\left[\frac{1}{2\sigma^2}w^T\Psi^T\Psi w - \frac{1}{\sigma^2}y^T\Psi w + \frac{\alpha}{2}w^T w\right]. \tag{5}$$

Although marginally convex on $w$, this objective is not jointly convex on $w$ and $\Pi$ (and involves non-convex constraints on $\Pi$ owing to its discreteness). The lack of joint convexity makes the optimization difficult. The typical approach in such situations is to use an alternating descent strategy, such as EM. Instead, in the following we develop a convex relaxation for problem (5).

# 3 Semidefinite Relaxation

To obtain a convex relaxation we proceed in three steps. First, we dualize the first term in (5).

**Lemma 1** *Define $A(\Psi w) := \frac{1}{2\sigma^2} w^T \Psi^T \Psi w$. Then the Fenchel dual of $A(\Psi w)$ is $A^*(c) = \frac{1}{2}\sigma^2 c^T c$, and therefore $A(\Psi w) = \max_c c^T \Psi w - \frac{1}{2}\sigma^2 c^T c$.*

**Proof** From the definition of Fenchel dual we have $A^*(u) := \max_w u^T w - \frac{1}{2\sigma^2} w^T \Psi^T \Psi w$. Differentiating with respect to $w$ and equating to zero we obtain $u = \frac{1}{\sigma^2} \Psi^T \Psi w$. Therefore $u$ is only realizable if there exists a $c$ such that $u = \Psi^T c$. Solving for $A^*(c)$ we obtain $A^*(c) = \frac{1}{2}\sigma^2 c^T c$, and therefore by definition of Fenchel duality $A(\Psi w) = \max_c c^T \Psi w - \frac{1}{2}\sigma^2 c^T c$. ∎

A second Lemma is required to further establish the relaxation:

**Lemma 2** *The following set inclusion holds*

$$\{\Pi\Pi^T : \Pi \in \{0,1\}^{t\times k}, \Pi\mathbf{1} = \mathbf{1}, \max(diag(\Pi^T\Pi)) \le \gamma t\} \tag{6}$$

$$\subseteq \{M : M \in \mathbb{R}^{t\times t}, \operatorname{tr} M = t, \gamma t I \succcurlyeq M \succcurlyeq 0\}. \tag{7}$$

**Proof** Let $\Pi\Pi^T$ be an element of the first set. First notice that $[\Pi\Pi^T]_{ij} \in \{0,1\}$ since $\Pi \in \{0,1\}^{t\times k}$ and $\Pi\mathbf{1} = \mathbf{1}$ together imply that $\Pi$ has a single 1 per row (and the rest are zeros). In particular $[\Pi\Pi^T]_{ii} = 1$ for all $i$, i.e. $\operatorname{tr} M = t$. Finally, note that $(\Pi\Pi^T)\Pi = \Pi(\Pi^T\Pi)$ where $\Pi^T\Pi$ is a diagonal matrix and therefore its diagonal elements are the eigenvalues of $\Pi\Pi^T$ and in particular $\max(diag(\Pi^T\Pi)) \le \gamma t$ means that the largest possible eigenvalue of $\Pi\Pi^T$ is $\gamma t$, which implies $\gamma t I \succcurlyeq \Pi\Pi^T$. Since $\Pi\Pi^T$ is by construction positive semidefinite, we have $\gamma t I \succcurlyeq \Pi\Pi^T \succcurlyeq 0$. Therefore $\Pi\Pi^T$ is also a member of the second set. ∎

The above two lemmas allow us to state our first main result below.

**Theorem 3** *The following convex optimization problem*

$$\min_{M:\operatorname{tr} M=t,\gamma t I \succcurlyeq M \succcurlyeq 0} \max_c \left[ -\frac{1}{2}\sigma^2 c^T c - \frac{1}{2\alpha}\left(\frac{y}{\sigma^2} - c\right)^T M \odot XX^T \left(\frac{y}{\sigma^2} - c\right)\right] \tag{8}$$

*is a relaxation of (5) only in the sense that domain (6) is replaced by domain (7).*

**Proof** We first use Lemma 1 in order to rewrite the objective (5) and obtain

$$\min_{\Pi,w} \left[\left(\max_c c^T \Psi w - \frac{1}{2}\sigma^2 c^T c\right) - \frac{1}{\sigma^2} y^T \Psi w + \frac{\alpha}{2} w^T w\right]. \tag{9}$$

Second, using the distributivity of the $(\max, +)$ semi-ring, the $\max_c$ can be pulled out and we then use Sion's minimax theorem [14], which allows us to interchange $\max_c$ with $\min_w$

$$\min_{\Pi} \max_c \min_w \left[c^T \Psi w - \frac{1}{2}\sigma^2 c^T c - \frac{1}{\sigma^2} y^T \Psi w + \frac{\alpha}{2} w^T w\right], \tag{10}$$

and we can solve for $w$ first, obtaining

$$w = \frac{1}{\alpha}\Psi^T\left(\frac{y}{\sigma^2} - c\right). \tag{11}$$

Substituting (11) in the objective of (10) results in

$$\min_{\Pi} \max_c \left[-\frac{1}{2}\sigma^2 c^T c - \frac{1}{2\alpha}\left(\frac{y}{\sigma^2} - c\right)^T \Psi\Psi^T\left(\frac{y}{\sigma^2} - c\right)\right]. \tag{12}$$

We now note the critical fact that $\Psi$ only shows up in the expression $\Psi\Psi^T$ which, from the definition $\psi_i = \pi_i \otimes x_i$, is seen to be equivalent to $\Pi\Pi^T \odot XX^T$. Therefore the minimization over $\Pi$ effectively takes place over $\Pi\Pi^T$ (since $X$ is observed), and we have that (12) can be rewritten as

$$\min_{\Pi\Pi^T} \max_c \left[-\frac{1}{2}\sigma^2 c^T c - \frac{1}{2\alpha}\left(\frac{y}{\sigma^2} - c\right)^T \Pi\Pi^T \odot XX^T \left(\frac{y}{\sigma^2} - c\right)\right]. \tag{13}$$

So far no relaxation has taken place. By finally replacing the constraint (6) with constraint (7) from Lemma 2, we obtain the claimed semidefinite relaxation. ∎

## 4  Max-Min Reformulation

By upper bounding the inner maximization in (8) and applying a Schur complement, problem (8) can be re-expressed as a semidefinite program. Unfortunately, such a formulation is computationally expensive to solve, requiring $O(t^6)$ for typical interior-point methods. Instead, we can reformulate problem (8) to allow for a fast algorithmic approach, without the introduction of any additional relaxation. The basis of our development is the following classical result.

**Theorem 4 ([15])** *Let $V \in \mathbb{R}^{t \times t}, V = V^T$ have eigenvalues $\lambda_1 \geq \lambda_2 \geq \cdots \geq \lambda_t$. Let $P$ be the matrix whose columns are the normalized eigenvectors of $V$, i.e. $P^T V P = \Lambda((\lambda_1, \ldots, \lambda_t))$. Let $q \in \{1, \ldots, t\}$ and $P_q$ be the matrix comprised by the top $q$ eigenvectors of $P$. Then*

$$\max_{M:\operatorname{tr}(M)=q, I \succcurlyeq M \succcurlyeq 0} \operatorname{tr} MV^T = \sum_{i=1}^{q} \lambda_i \qquad and \qquad (14)$$

$$\operatorname*{argmax}_{M:\operatorname{tr}(M)=q, I \succcurlyeq M \succcurlyeq 0} \operatorname{tr} MV^T \ni P_q P_q^T. \qquad (15)$$

**Proof** See [15] for a proof of a slightly more general result (Theorem 3.4). ∎

We will now show how the optimization on $M$ for problem (8) can be cast in the terms of Theorem 4. This will turn out to be critical for the efficiency of the optimization procedure, since Theorem 4 describes a purely spectral optimization routine, which is far more efficient ($O(t^3)$) than standard interior-point methods used for semidefinite programming ($O(t^6)$).

**Proposition 5** *Define $\bar{y} := \frac{y}{\sigma^2}$. The following optimization problem*

$$\max_{c} \left[ -\frac{1}{2}\sigma^2 c^T c - \frac{1}{2\alpha} \max_{M:\operatorname{tr} M=t, \gamma t I \succcurlyeq M \succcurlyeq 0} \operatorname{tr}(M(XX^T \odot (\bar{y}-c)(\bar{y}-c)^T)) \right] \qquad (16)$$

*is equivalent to optimization problem (8).*

**Proof** By Sion's minimax theorem [14], $\min_M$ and $\max_c$ in (8) can be interchanged

$$\max_{c} \min_{M:\operatorname{tr} M=t, \gamma t I \succcurlyeq M \succcurlyeq 0} \left[ -\frac{1}{2}\sigma^2 c^T c - \frac{1}{2\alpha} (\bar{y}-c)^T M \odot XX^T (\bar{y}-c) \right] \qquad (17)$$

which, by distributivity of the $(\min, +)$ semi-ring, is equivalent to

$$\max_{c} \left[ -\frac{1}{2}\sigma^2 c^T c + \frac{1}{2\alpha} \min_{M:\operatorname{tr} M=t, \gamma t I \succcurlyeq M \succcurlyeq 0} -(\bar{y}-c)^T M \odot XX^T (\bar{y}-c) \right]. \qquad (18)$$

Now, define $K := XX^T$. The objective of the minimization in (18) can then be written as

$$-(\bar{y}-c)^T (M \odot K)(\bar{y}-c) = -\operatorname{tr} \left( (M \odot K) \left[ (\bar{y}-c)(\bar{y}-c)^T \right] \right) \qquad (19)$$

$$= -\sum_{ij} (M_{ij} K_{ij}) \left[ (\bar{y}-c)(\bar{y}-c)^T \right]_{ij} = -\sum_{ij} M_{ij} \left( K_{ij} \left[ (\bar{y}-c)(\bar{y}-c)^T \right]_{ij} \right) \qquad (20)$$

$$= -\operatorname{tr}(M(K \odot (\bar{y}-c)(\bar{y}-c)^T)) = -\operatorname{tr}(M(XX^T \odot (\bar{y}-c)(\bar{y}-c)^T)). \qquad (21)$$

Finally, by writing $\min_M -f(M)$ as $-\max_M f(M)$, we obtain the claim. ∎

We can now exploit the result in Theorem 4 for the purpose of our optimization problem.

**Proposition 6** *Let $q = \{u : u = \max\{1, \ldots, t\}, u \leq \gamma^{-1}\}$. The following optimization problem*

$$\max_{c} \left[ -\frac{1}{2}\sigma^2 c^T c - \frac{t}{2\alpha q} \max_{\bar{M}:\operatorname{tr} \bar{M}=q, I \succcurlyeq \bar{M} \succcurlyeq 0} \operatorname{tr}(\bar{M}(XX^T \odot (\bar{y}-c)(\bar{y}-c)^T)) \right] \qquad (22)$$

*is equivalent to optimization problem (16).*

**Algorithm 1**

1: **Input:** $\gamma, \sigma, \alpha, XX^T$
2: **Output:** $(c^*, M^*)$
3: Initialize $c = 0$
4: **repeat**
5:     Solve for maximum value in inner maximization of (22) using (14)
6:     Solve outer maximization in (22) using nonsmooth BFGS [16], obtain new $c$
7: **until** $c$ has converged ($c = c^*$)
8: At $c^*$, solve for the maximizer(s) $P_q$ in the inner maximization of (22) using (15)
9: **if** $P_q$ is unique **then**
10:     **return** $M^* = P_q P_q^T$ **break**
11: **else**
12:     Assemble top $l$ eigenvectors in $P_l$
13:     Solve (24)
14:     **return** $M^* = P_l \Lambda(\lambda^*) P_l^T$
15: **end if**

**Proof** The only differences between (16) and (22) are (i) the factor $t/q$ in the second term of (22) and (ii) the constraints $\{M : \operatorname{tr} M = t, \gamma t I \succcurlyeq M \succcurlyeq 0\}$ in (16) versus $\{M : \operatorname{tr} M = q, I \succcurlyeq M \succcurlyeq 0\}$ in (22). These differences are simply the result of a proper rescaling of $M$. If we define $\bar{M} := (q/t)M$, then $I \succcurlyeq \bar{M} \succcurlyeq 0$ since $q \leq \gamma^{-1}$. We then have $\operatorname{tr} \bar{M} = q$. The result follows.   ∎

And finally we have the second main result

**Theorem 7** *Optimization problem (22) is equivalent to optimization problem (8).*

**Proof** The equivalence follows directly from Propositions 5 and 6.   ∎

Note that, crucially, the objective in (22) is *concave* in $c$. Our strategy is now clear. Instead of solving (8), which demands $O(t^6)$ operations, we instead solve (22), which has as inner optimization a max eigenvalue problem, demanding only $O(t^3)$ operations. In the next section we describe an algorithm to jointly optimize for $M$ and $c$ in (22), which will essentially consist of alternating the efficient spectral solution over $M$ with a subgradient optimization over $c$.

### 4.1 Max-Min Algorithm

Algorithm 1 describes how we solve optimization problem (22). The idea of the algorithm is the following. First, having noted that (22) is concave in $c$, we can simply initialize $c$ arbitrarily and pursue a fast subgradient ascent algorithm (e.g. such as nonsmooth BFGS [16]). So at each step we solve the eigenvalue problem and recompute a subgradient, until convergence to $c^*$. We then need to recover $M^*$ such that $(c^*, M^*)$ is a saddle point (note that problem (22) is concave in $c$ and convex in $M$). For that purpose we use (15). If $M^* = P_q P_q^T$ is such that $P_q$ is unique, then we are done and the labeling solution of mixture membership is $M^*$ (subject to roundoff). If $P_q$ is not unique, then we have multiplicity of eigenvalues and we need to proceed as follows. Define $P_l = [p_1 \ldots p_q \ldots p_l], l > q$, where each of the additional $l - q$ eigenvectors has an associated eigenvalue which is equal to the eigenvalue of some of the previous $q$ eigenvectors. We then have that at the saddle point there must exist a diagonal matrix $\Lambda$ such that $M^* = P_l \Lambda P_l^T$, subject to $\Lambda \succcurlyeq 0$ and $\operatorname{tr} \Lambda = q$ (if this were not the case there would be an ascent direction in $c^*$, contradicting the hypothesis that $c^*$ is optimal). To find such a $\Lambda$ and therefore recover the correct $M$, we need to enforce that we are at the optimal $c$ ($c^*$), i.e. we must have

$$\left\| \frac{d}{dc} \left[ -\frac{1}{2} \sigma^2 c^T c - \frac{q}{2\alpha t} \max_{M : \operatorname{tr} M = q, I \succcurlyeq M \succcurlyeq 0} \operatorname{tr}(M(XX^T \odot (\bar{y} - c)(\bar{y} - c)^T)) \right] \right\|_2^2 = 0 \qquad (23)$$

Such condition can be pursued by minimizing the above norm, which gives a quadratic program

$$\min_{\lambda \geq 0, \lambda^T \mathbf{1} = q} \left\| \sigma^2 c^* + \frac{q}{\alpha t} \left( P_l \Lambda(\lambda) P_l^T \odot XX^T \right) (c^* - \bar{y}) \right\|_2^2 \qquad (24)$$

We can then recover the final solution (subject to roundoff) by $M^* = P_l \Lambda(\lambda^*) P_l^T$, where $\lambda^*$ is the optimizer of (24). The optimal value of (24) should be very close to zero (since it's the norm of the derivative at point $c^*$). The pseudocode for the algorithm appears in Algorithm 1.

---

**Algorithm 2**

---

1: **Input:** $\gamma, \sigma, \alpha, XX^T$
2: **Output:** $(c^*, M^*)$
3: Initialize $M = \Lambda((1/(\gamma t), \ldots, 1/(\gamma t)))$
4: **repeat**
5:     Solve for minimum value in inner minimization of (25), obtain $A$
6:     Solve outer minimization in (25) given SVD of $A$ using Theorem 4.1 of [18], obtain new $M$
7: **until** $M$ has converged $(M = M^*)$
8: Recover $c^* = \frac{-1}{\sigma^2} \text{diag}(X(A^*)^T)$

---

## 5 Min-Min Reformulation

Although the max-min formulation appears satisfactory, the recent literature on multitask learning [17, 18] has developed an alternate strategy for bypassing general semidefinite programming. Specifically, work in this area lead to convex optimization problems expressed jointly over two matrix variables where each step is an alternating min-min descent that can be executed in closed-form or by a very fast algorithm. Although it is not immediately apparent that this algorithmic strategy is applicable to the problem at hand, with some further reformulation of (8) we discover that in fact the same min-min algorithmic approach can be applied to our mixture of regression problem.

**Theorem 8** *The following optimization problem*

$$\min_{\{M:I\succeq M\succeq 0,\text{tr }M=1/\gamma\}} \min_A \left[ \frac{1}{\sigma^2} y^T \text{diag}(XA^T) + \frac{1}{2\sigma^2} \text{diag}(XA^T)^T \text{diag}(XA^T) + \frac{\alpha}{2\gamma t} \text{tr}(A^T M^{-1} A) \right] \tag{25}$$

*is equivalent to optimization problem (8).*

**Proof**

$$\min_{\{M:I\succeq M\succeq 0,\text{tr }M=1/\gamma\}} \max_c -\frac{\sigma^2}{2} c^T c - \frac{\gamma t}{2\alpha}(c-\bar{y})^T (M \odot XX^T)(c-\bar{y}) \tag{26}$$

$$= \min_{\{M:I\succeq M\succeq 0,\text{tr }M=1/\gamma\}} \max_{\{c,C:C=\Lambda(c-\bar{y})X\}} -\frac{\sigma^2}{2} c^T c - \frac{\gamma t}{2\alpha} \text{tr}(C^T MC) \tag{27}$$

$$= \min_{\{M:I\succeq M\succeq 0,\text{tr }M=1/\gamma\}} \min_A \max_{c,C} -\frac{\sigma^2}{2} c^T c - \frac{\gamma t}{2\alpha} \text{tr}(C^T MC) + \text{tr}(A^T C) - \text{tr}(A^T \Lambda(c-\bar{y})X) \tag{28}$$

We can then solve for $c$ and $C$, obtaining $c = -\frac{1}{\sigma^2}\text{diag}(XA^T)$ and $C = \frac{\alpha}{\gamma t}M^{-1}A$. Substituting those two variables into (28) proves the claim. ∎

### 5.1 Min-Min Algorithm

The problem (25) is jointly convex in $A$ and $M$ [14] and Algorithm 2 describes how to solve it. It is important to note that although each iteration in Algorithm 2 is efficient, many iterations are required to reach a desired tolerance, since it is only first-order convergent. It is observed in our experiments that the concave-convex max-min approach in Algorithm 1 is more efficient simply because it has the same iteration cost but exploits a quasi-Newton descent in the outer optimization, which converges faster.

**Remark 9** *In practice, similarly to [17], a regularizer on $M$ is added to avoid singularity, resulting in the following regularized objective function,*

$$\min_{\{M:I\succeq M\succeq 0,\text{tr }M=1/\gamma\}} \quad \min_A \frac{1}{\sigma^2} y^T \text{diag}(XA^T) + \frac{1}{2\sigma^2} \text{diag}(XA^T)^T \text{diag}(XA^T)$$

$$+ \frac{\alpha}{2\gamma t} \text{tr}(A^T M^{-1} A) + \epsilon \, \text{tr}(M^{-1}). \tag{29}$$

*The problem is still jointly convex in $M$ and $A$.*

# 6 Experiments

Our primary objective in formulating this convex approach to mixture regression is to tackle a difficult problem in video analysis (see below). However, to initially evaluate the different approaches we conducted some synthetic experiments. We generated 30 synthetic data points according to $y_i = (\pi_i \otimes x_i)w + \epsilon_i$, with $x_i \in \mathbb{R}$, $\epsilon_i \sim N(0,1)$ and $w \in U(0,1)$. The response variable $y_i$ is assumed to be generated from a mixture of 5 components. We compared the quality of the relaxation in (22) to EM. Max-min algorithm is used in this experiment. For EM, 100 random restarts was used to help avoid poor local optima. The experiment is repeated 10 times. The error rates are $0.347 \pm 0.086$ and $0.280 \pm 0.063$ for EM and convex relaxation, respectively. The visualization of the recovered membership for one of the runs is given in Figure 1. This demonstrates that the relaxation can retain much of the structure of the problem.

## 6.1 Vision Experiment

In a dynamic scene, various static and moving objects are viewed by a possibly moving observer. For example, consider a moving, hand-held camera filming a scene of several cars driving down the road. Each car has a separate motion, and even the static objects, such as trees, appear to move in the video due to the self-motion of the camera. The task of segmenting each object according to its motion, estimating the parameters of each motion, and recovering the structure of the scene is known as the *multibody structure and motion problem*. This is a missing variable problem. If the motions have been segmented correctly, it is easy to estimate the parameters of each motion. Naturally, models employing EM have been proposed to tackle such problems (c.f. [19, 20]).

From epipolar geometry, given a pair of corresponding points $p_i$ and $q_i$ from two images ($p_i, q_i \in \mathbb{R}^{3 \times 1}$), we have the epipolar equation $q_i^T F p_i = 0$. The fundamental matrix $F$ encapsulates information about the translation and rotation relative to the scene points between the positions of the camera where the two images were captured, as well as the camera calibration parameters such as its focal length. In a static scene, where only the camera is moving, there is only one fundamental matrix, which arises from the camera self-motion. However, if some of the scene points are moving independently under multiple different motions, there are several fundamental matrices. If there are $k$ motion groups, the epipolar equation can be expressed in term of the multibody fundamental matrix [21], i.e. $\prod_{j=1}^{k}(q_i^T F_j p_i) = 0$. An algebraic method was proposed to recover this matrix via Generalized PCA [21]. An alternative approach, which we follow here, is by Li [22], who casts the problem as a mixture of fundamental matrices, i.e. $q_i^T (\sum_{j=1}^{k} \pi_{ij} F_j) p_i = 0$ where the membership variable $\pi_{ij} = 1$ when image point $i$ belongs to motion group $j$, and zero otherwise. Furthermore, since $q_i^T F p_i = 0$ is bilinear in the image points, we can rewrite it to be $x_i^T w_j = 0$, with the column vectors $x_i = [q_i^x p_i^x \quad q_i^x p_i^y \quad q_i^x p_i^\pi \quad .... \quad q_i^\pi p_i^\pi]^T$ and $w = \text{vec}(F_j^T)$. Thus, we will end up with the following linear equation: $\sum_{j=1}^{k} \pi_{ij} x_i^T w_j = 0$. The weight vector $w_j$ for motion group $j$ can be recovered easily if the indicator variable $\pi_{ij}$ is known.

We are interested in assessing the effectiveness of EM-based and convex relaxation-based methods for this multibody structure and motion problem. We used the Hopkins 155 dataset [23]. The experimental results are summarized in Table 1. All hyperparameters (EM: $\alpha$ and $\sigma$; Convex relaxation: $\alpha$, $\sigma$, and $\gamma$) were tuned and the best performances for each learning algorithm are reported. The EM algorithm was run with 100 random restarts to help avoid poor local optima. In terms of computation time, the max-min runs comparably to the EM algorithm, while min-min runs in the order of 3 to 4 times slower. As an illustration, on a Pentium 4 3.6 GHz machine, the elapsed time (in seconds) for `two cranes` dataset is 16.880, 23.536, and 60.003 for EM, max-min and min-min, respectively. Rounding for the convex versions was done by k-means, which introduces some differences in the final results for both algorithms. Noticeably, both max-min and min-min outperform the EM algorithm. Visualizations of the motion segmentation on `two cranes`, `three cars`, and `cars2_07` datasets are given in Figure 2 (for `kanatani2` and `articulated` please refer to Appendix).

# 7 Conclusion

The mixture regression problem is pervasive in many applications and known approaches for parameter estimation rely on variants of EM, which naturally have issues with local minima. In this paper we introduced a semidefinite relaxation for the mixture regression problem, thus obtaining a convex formulation which does not suffer from local minima. In addition we showed how to avoid the

use of expensive interior-point methods typically needed to solve semidefinite programs. This was achieved by introducing two reformulations amenable to the use of faster algorithms. Experimental results with synthetic data as well as with real computer vision data suggest the proposed methods can substantially improve on EM while one of the methods in addition has comparable runtimes.

Table 1: Error rate on several datasets from the Hopkins 155

| Data set | $m$ | EM | Max-Min Convex | Min-Min Convex |
|---|---|---|---|---|
| three cars | 173 | 0.0532 | 0.0289 | 0.0347 |
| kanatani2 | 63 | 0.0000 | 0.0000 | 0.0000 |
| cars2_07 | 212 | 0.3396 | 0.2642 | 0.2594 |
| two cranes | 94 | 0.0532 | 0.0213 | 0.0106 |
| articulated | 150 | 0.0000 | 0.0000 | 0.0000 |

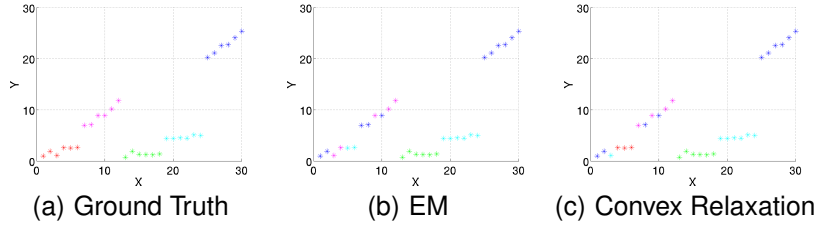

(a) Ground Truth     (b) EM     (c) Convex Relaxation

Figure 1: Recovered membership on synthetic data with EM and convex relaxation. 30 data points are generated according to $y_i = (\pi_i \otimes x_i)w + \epsilon_i$, with $x_i \in \mathbb{R}$, $\epsilon_i \sim N(0,1)$ and $w \in U(0,1)$.

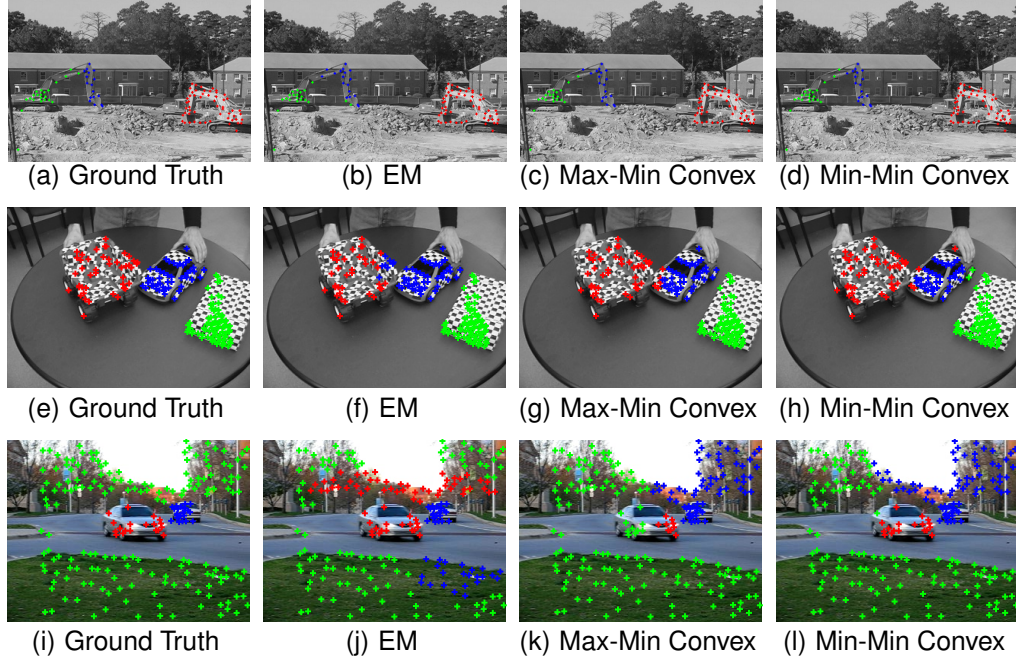

(a) Ground Truth    (b) EM    (c) Max-Min Convex    (d) Min-Min Convex

(e) Ground Truth    (f) EM    (g) Max-Min Convex    (h) Min-Min Convex

(i) Ground Truth    (j) EM    (k) Max-Min Convex    (l) Min-Min Convex

Figure 2: Resulting motion segmentations produced by the various techniques on the Hopkins 155 dataset. 2(a)-2(d): two cranes, 2(e)-2(h): three cars, and 2(i)-2(l): cars2_07. In two cranes (first row), EM produces more segmentation errors at the left crane. In three cars (second row), the max-min method gives the least segmentation error (at the front side of the middle car) and EM produces more segmentation errors at the front side of the left car. The contrast of EM and convex methods is apparent for cars2_07 (third row): the convex methods segment correctly the static grass field object, while EM makes mistakes. Further, the min-min method can almost perfectly segment the car in the middle of the scene from the static tree background.

# References

[1] S. Gaffney and P. Smyth. Trajectory clustering with mixtures of regression models. In *ACM SIGKDD*, volume 62, pages 63–72, 1999.

[2] S. Finney, L. Kaelbling, and T. Lozano-Perez. Predicting partial paths from planning problem parameters. In *Proceedings of Robotics: Science and Systems*, Atlanta, GA, USA, June 2007.

[3] A. J. Storkey and M. Sugiyama. Mixture regression for covariate shift. In Schölkopf, editor, *Advances in Neural Information Processing Systems 19*, pages 1337–1344, 2007.

[4] A. P. Dempster, N. M. Laird, and D. B. Rubin. Maximum likelihood from incomplete data via the em algorithm. *Journal of the Royal Statistical Society. Series B (Methodological)*, 39(1):1–38, 1977.

[5] T. De Bie, N. Cristianini, P. Bennett, and E. Parrado-hernández. Fast sdp relaxations of graph cut clustering, transduction, and other combinatorial problems. *JMLR*, 7:1409–1436, 2006.

[6] Y. Guo and D. Schuurmans. Convex relaxations for latent variable training. In Platt et al., editor, *Advances in Neural Information Processing Systems 20*, pages 601–608, 2008.

[7] S. M. Goldfeld and R.E. Quandt. Nonlinear methods in econometrics. *Amsterdam: North-Holland Publishing Co.*, 1972.

[8] D. W. Hosmer. Maximum likelihood estimates of the parameters of a mixture of two regression lines. *Communications in Statistics*, 3(10):995–1006, 1974.

[9] H. Späth. Algorithm 39: clusterwise linear regression. *Computing*, 22:367–373, 1979.

[10] W.S. DeSarbo and W.L. Cron. A maximum likelihood methodology for clusterwise linear regression. *Journal of Classification*, 5(1):249–282, 1988.

[11] P.N. Jones and G.J. McLachlan. Fitting finite mixtures models in a regression context. *Austral. J. Statistics*, 34(2):233–240, 1992.

[12] S. Gaffney and P. Smyth. Curve clustering with random effects regression mixtures. In *AIS-TATS*, 2003.

[13] M.I. Jordan and R.A. Jacobs. Hierarchical mixtures of experts and the em algorithm. *Neural computation*, 6:181–214, 1994.

[14] S. Boyd and L. Vandenberghe. *Convex Optimization*. Cambridge University Press, 2004.

[15] M. Overton and R. Womersley. Optimality conditions and duality theory for minimizing sums of the largest eigenvalues of symmetric matrices. *Mathematical Programming*, 62:321–357, 1993.

[16] J. Yu, S.V.N. Vishwanathan, S. Günter, and N. Schraudolph. A quasi-Newton approach to nonsmooth convex optimization. In *ICML*, 2008.

[17] A. Argyriou, T. Evgeniou, and M. Pontil. Convex multi-task feature learning. *Machine Learning*, 73:243–272, 2008.

[18] J. Chen, L. Tang, J. Liu, and J. Ye. A convex formulation for learning shared structures from multiple tasks. In *ICML*, 2009.

[19] N.Vasconcelos and A. Lippman. Empirical bayesian em-based motion segmentation. In *CVPR*, 1997.

[20] P. Torr. Geometric motion segmentation and model selection. *Philosophical Trans. of the Royal Society of London*, 356(1740):1321–1340, 1998.

[21] R. Vidal, Y. Ma, S. Soatto, and S. Sastry. Two-view multibody structure from motion. *IJCV*, 68(1):7–25, 2006.

[22] H. Li. Two-view motion segmentation from linear programming relaxation. In *CVPR*, 2007.

[23] http://www.vision.jhu.edu/data/hopkins155/.

